# Handling Advertisements of Unknown Quality in Search Advertising

**Sandeep Pandey**
**Carnegie Mellon University**
spandey@cs.cmu.edu

**Christopher Olston**
**Yahoo! Research**
olston@yahoo-inc.com

## Abstract

We consider how a search engine should select advertisements to display with search results, in order to maximize its revenue. Under the standard "pay-per-click" arrangement, revenue depends on how well the displayed advertisements appeal to users. The main difficulty stems from new advertisements whose degree of appeal has yet to be determined. Often the only reliable way of determining appeal is *exploration* via display to users, which detracts from *exploitation* of other advertisements known to have high appeal. Budget constraints and finite advertisement lifetimes make it necessary to explore as well as exploit.

In this paper we study the tradeoff between exploration and exploitation, modeling advertisement placement as a multi-armed bandit problem. We extend traditional bandit formulations to account for budget constraints that occur in search engine advertising markets, and derive theoretical bounds on the performance of a family of algorithms. We measure empirical performance via extensive experiments over real-world data.

## 1 Introduction

Search engines are invaluable tools for society. Their operation is supported in large part through advertising revenue. Under the standard "pay-per-click" arrangement, search engines earn revenue by displaying appealing advertisements that attract user clicks. Users benefit as well from this arrangement, especially when searching for commercial goods or services.

Successful advertisement placement relies on knowing the appeal or "clickability" of advertisements. The main difficulty is that the appeal of new advertisements that have not yet been "vetted" by users can be difficult to estimate. In this paper we study the problem of placing advertisements to maximize a search engine's revenue, in the presence of uncertainty about appeal.

### 1.1 The Advertisement Problem

Consider the following *advertisement problem* [8], illustrated in Figure 1. There are $m$ advertisers $A_1, A_2 \ldots A_m$ who wish to advertise on a search engine. The search engine runs a large auction where each advertiser submits its bids to the search engine for the query phrases in which it is interested. Advertiser $A_i$ submits advertisement $a_{i,j}$ to *target* query phrase $Q_j$, and promises to pay $b_{i,j}$ amount of money for each click on this advertisement, where $b_{i,j}$ is $A_i$'s bid for advertisement $a_{i,j}$. Advertiser $A_i$ can also specify a daily budget $(d_i)$ that is the total amount of money it is willing to pay for the clicks on its advertisements in a day. Given a user search query on phrase $Q_j$, the search engine selects a constant number $C \geq 1$ of advertisements from the candidate set of advertisements $\{a_{*,j}\}$, targeted to $Q_j$. The objective in selecting advertisements is to maximize the search engine's total daily revenue. The arrival sequence of user queries is not known in advance. For now we assume that each day a new set of advertisements is given to the search engine and the set remains fixed through out the day; we drop both of these assumptions later in Section 4.

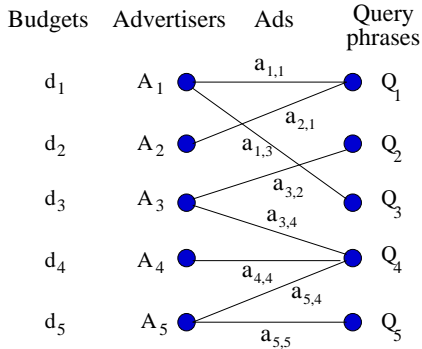

| Budgets | Advertisers | Ads | Query phrases |
|---|---|---|---|

Figure 1: Advertiser and query model.

| | CTR = 1 for all ads | general CTR, CTR known | general CTR, CTR unknown |
|---|---|---|---|
| no budget constraints | I<br>GREEDY<br>ratio=1 | III<br>GREEDY<br>ratio=1 | V<br>this paper |
| budget constraints | II<br>MSVV<br>ratio=1−1/e | IV<br>GREEDY<br>ratio=1/2 | VI<br>this paper |

Figure 2: Problem variants.

High revenue is achieved by displaying advertisements that have high bids as well as high likelihood of being clicked on by users. Formally, the *click-through rate* (CTR) $c_{i,j}$ of advertisement $a_{i,j}$ is the probability of a user to click on advertisement $a_{i,j}$ given that the advertisement was displayed to the user for query phrase $Q_j$. In the absence of budget constraints, revenue is maximized by displaying advertisements with the highest $c_{i,j} \cdot b_{i,j}$ value. The work of [8] showed how to maximize revenue in the presence of budget constraints, but under the assumption that all CTRs are known in advance. In this paper we tackle the more difficult but realistic problem of maximizing advertisement revenue when CTRs are not necessarily known at the outset, and must be learned on the fly.

We show the space of problem variants (along with the best known advertisement policies) in Figure 2. GREEDY refers to selection of advertisements according to expected revenue (*i.e.*, $c_{i,j} \cdot b_{i,j}$). In Cells I and III GREEDY performs as well as the optimal policy, where the optimal policy also knows the arrival sequence of queries in advance. We write "ratio=1" in Figure 2 to indicate that GREEDY has the competitive ratio of 1. For Cells II and IV the greedy policy is not optimal, but is nevertheless 1/2 competitive. An alternative policy for Cell II was given in [8], which we refer to as MSVV; it achieves a competitive ratio of $1 - 1/e$. In this paper we give the first policies for Cells V and VI, where we must choose which advertisements to display while simultaneously estimating click-through rates of advertisements.

## 1.2 Exploration/Exploitation Tradeoff

The main issue we face while addressing Cells V and VI is to balance the exploration/exploitation tradeoff. To maximize short-term revenue, the search engine should *exploit* its current, imperfect CTR estimates by displaying advertisements whose estimated CTRs are large. On the other hand, to maximize long-term revenue, the search engine needs to *explore*, *i.e.*, identify which advertisements have the largest CTRs. This kind of exploration entails displaying advertisements whose current CTR estimates are of low confidence, which inevitably leads to displaying some low-CTR ads in the short-term. This kind of tradeoff between *exploration* and *exploitation* shows up often in practice, e.g., in clinical trials, and has been extensively studied in the context of the *multi-armed bandit* problem [4].

In this paper we draw upon and extend the existing bandit literature to solve the advertisement problem in the case of unknown CTR. In particular, first in Section 3 we show that the unbudgeted variant of the problem (Cell V in Figure 2) is an instance of the multi-armed bandit problem. Then, in Section 4 we introduce a new kind of bandit problem that we termed the *budgeted multi-armed multi-bandit* problem (BMMP), and show that the budgeted unknown-CTR advertisement problem (Cell VI) is an instance of BMMP. We propose policies for BMMP and give performance bounds. We evaluate our policies empirically over real-world data in Section 5. In the extended technical version of the paper [9] we show how to extend our policies to address various practical considerations, *e.g.*, exploiting any prior information available about the CTRs of ads, permitting advertisers to submit and revoke advertisements at any time, not just at day boundaries.

## 2  Related Work

We have already discussed the work of [8], which addresses the advertisement problem under the assumption that CTRs are known. There has not been much published work on estimating CTRs. Reference [7] discusses how contextual information such as user demographic or ad topic can be used to estimate CTRs, and makes connections to the recommender and bandit problems, but stops short of presenting technical solutions. Some methods for estimating CTRs are proposed in [5] with the focus of thwarting click fraud.

Reference [1] studies how to maximize user clicks on banner ads. The key problem addressed in [1] is to satisfy the contracts made with the advertisers in terms of the minimum guaranteed number of impressions (as opposed to the budget constraints in our problem). Reference [10] looks at the advertisement problem from an advertiser's point of view, and gives an algorithm for identifying the most profitable set of keywords for the advertiser.

## 3  Unbudgeted Unknown-CTR Advertisement Problem

In this section we address Cell V of Figure 2, where click-through rates are initially unknown and budget constraints are absent (*i.e., $d_i = \infty$* for all advertisers $A_i$). Our unbudgeted problem is an instance of the multi-armed bandit problem [4], which is the following: we have $K$ *arms* where each arm has an associated reward and payoff probability. The payoff probability is not known to us while the reward may or may not be known (both versions of the bandit problem exist). With each *invocation* we *activate* exactly $C \leq K$ arms. [1] Each activated arm then yields the associated reward with its payoff probability and nothing with the remaining probability. The objective is to determine a *policy* for activating the arms so as to maximize the total reward over some number of invocations.

To solve the unbudgeted unknown-CTR advertisement problem, we create a multi-armed bandit problem instance for each query phrase $Q$, where ads targeted for the query phrase are the arms, bid values are the rewards and CTRs are the payoff probabilities of the bandit instance. Since there are no budget constraints, we can treat each query phrase independently and solve each bandit instance in isolation. [2] The number of invocations for a bandit instance is not known in advance because the number of queries of phrase $Q$ in a given day is not known in advance.

A variety of policies have been proposed for the bandit problem, *e.g.,* [2, 3, 6], any of which can be applied to our unbudgeted advertisement problem. The policies proposed in [3] are particularly attractive because they have a known performance bound for any number of invocations not known in advance (in our context the number of queries is not known a priori). In the case of $C = 1$, the policies of [3] make $O(ln\ n)$ number of *mistakes*, on expectation, in $n$ invocations (which is also the asymptotic lower bound on the number of mistakes [6]). A mistake occurs when a suboptimal arm is chosen by a policy (the optimal arm is the one with the highest expected reward).

We consider a specific policy from [3] called UCB and apply it to our problem (other policies from [3] can also be used). UCB is proposed under a slightly different reward model; we adapt it to our context to produce the following policy that we call *MIX* (for mixing exploration with exploitation). We prove a performance bound of $O(ln\ n)$ mistakes for MIX for any $C \geq 1$ in [9].

**Policy MIX :**
 *Each time a query for phrase $Q_j$ arrives:*

  1. *Display the $C$ ads targeted for $Q_j$ that have the highest priority. The priority $P_{i,j}$ of ad $a_{i,j}$ is a function of its current CTR estimate ($\hat{c}_{i,j}$), its bid value ($b_{i,j}$), the number of times it has been displayed so far ($n_{i,j}$), and the number of times phrase $Q_j$ has been queried so far in the day ($n_j$). Formally, priority $P_{i,j}$ is defined as:*

$$P_{i,j} = \begin{cases} \left( \hat{c}_{i,j} + \sqrt{\frac{2\ ln\ n_j}{n_{i,j}}} \right) \cdot b_{i,j} & \text{if } n_{i,j} > 0 \\ \infty & \text{otherwise} \end{cases}$$

2. *Monitor the clicks made by users and update the CTR estimates $\hat{c}_{i,j}$ accordingly. $\hat{c}_{i,j}$ is the average click-through rate observed so far, i.e., the number of times ad $a_{i,j}$ has been clicked on divided by the total number of times it has been displayed.*

Policy MIX manages the exploration/exploitation tradeoff in the following way. The priority function has two factors: an exploration factor $\left(\sqrt{\frac{2\ ln\ n_j}{n_{i,j}}}\right)$ that diminishes with time, and an exploitation factor $(\hat{c}_{i,j})$. Since $\hat{c}_{i,j}$ can be estimated only when $n_{i,j} \geq 1$, the priority value is set to $\infty$ for an ad which has never been displayed before.

Importantly, the MIX policy is practical to implement because it can be evaluated efficiently using a single pass over the ads targeted for a query phrase. Furthermore, it incurs minimal storage overhead because it keeps only three numbers ($\hat{c}_{i,j}$, $n_{i,j}$ and $b_{i,j}$) with each ad and one number ($n_j$) with each query phrase.

## 4  Budgeted Unknown-CTR Advertisement Problem

We now turn to the more challenging case in which advertisers can specify daily budgets (Cell VI of Figure 2). Recall from Section 3 that in the absence of budget constraints, we were able to treat the bandit instance created for a query phrase independent of the other bandit instances. However, budget constraints create dependencies between query phrases targeted by an advertiser. To model this situation, we introduce a new kind of bandit problem that we call **B**udgeted **M**ulti-armed **M**ulti-bandit **P**roblem (BMMP), in which multiple bandit instances are run in parallel under overarching budget constraints. We derive generic policies for BMMP and give performance bounds.

### 4.1  Budgeted Multi-armed Multi-bandit Problem

BMMP consists of a finite set of multi-armed bandit instances, $\mathcal{B} = \{B_1, B_2 \ldots B_{|\mathcal{B}|}\}$. Each bandit instance $B_i$ has a finite number of arms and associated rewards and payoff probabilities as described in Section 3. In BMMP each arm also has an associated *type*. Each type $T_i \in \mathcal{T}$ has budget $d_i \in [0, \infty]$ which specifies the maximum amount of reward that can be generated by activating all the arms of that type. Once the specified budget is reached for a type, the corresponding arms can still be activated but no further reward is earned.

With each invocation of the bandit system, one bandit instance from $\mathcal{B}$ is invoked; the policy has no control over which bandit instance is invoked. Then the policy activates $C$ arms of the invoked bandit instance, and the activated arms generate some (possibly zero) total reward.

It is easy to see that the budgeted unknown-CTR advertisement problem is an instance of BMMP. Each query phrase acts as a bandit instance and the ads targeted for it act as bandit arms, as described in Section 3. Each advertiser defines a unique type of arms and gives a budget constraint for that type; all ads submitted by an advertiser belong to the type defined by it. When a query is submitted by a user, the corresponding bandit instance is invoked.

We now show how to derive a policy for BMMP given as input a policy POL for the regular multi-armed bandit problem such as one of the policies from [3]. The derived policy, denoted by *BPOL* (**B**udget-aware POL), is as follows:

- Run $|\mathcal{B}|$ instances of POL in parallel, denoted $\text{POL}_1, \text{POL}_2, \ldots \text{POL}_{|\mathcal{B}|}$.
- Whenever bandit instance $B_i$ is invoked:
  1. Discard any arm(s) of $B_i$ whose type's budget is newly depleted, *i.e.*, has become depleted since the last invocation of $B_i$.
  2. If one or more arms of $B_i$ was discarded during step 1, restart $\text{POL}_i$.
  3. Let $\text{POL}_i$ decide which of the remaining arms of $B_i$ to activate.

Observe that in the second step of BPOL, when POL is restarted, POL loses any state it has built up, including any knowledge gained about the payoff probabilities of bandit arms. Surprisingly, despite this seemingly imprudent behavior, we can still derive a good performance bound for BPOL, provided that POL has certain properties, as we discuss in

the next section. In practice, since most bandit policies can take prior information about the payoff probabilities as input, when restarting POL we can supply the previous payoff probability estimates as the prior (as done in our experiments).

## 4.2 Performance Bound for BMMP Policies

Let $S$ denote the sequence of bandit instances that are invoked, *i.e.*, $S = \{S(1), S(2) \ldots S(N)\}$ where $S(n)$ denotes the index of the bandit instance invoked at the $n^{\text{th}}$ invocation. We compare the performance of BPOL with that of the optimal policy, denoted by OPT, where OPT has advance knowledge of $S$ and the exact payoff probabilities of all bandit instances.

We claim that $bpol(N) \geq opt(N)/2 - O(f(N))$ for any $N$, where $bpol(N)$ and $opt(N)$ denote the total expected reward obtained after $N$ invocations by BPOL and OPT, respectively, and $f(n)$ denotes the expected number of mistakes made by POL after $n$ invocations of the the regular multi-armed bandit problem (for UCB, $f(n)$ is $O(ln\ n)$ [3]). Our complete proof is rather involved. Here we give a high-level outline of the proof (the complete proof is given in [9]). For simplicity we focus on the $C = 1$ case; $C \geq 1$ is a simple extension thereof.

Since bandit arms generate rewards stochastically, it is not clear how we should compare BPOL and OPT. For example, even if BPOL and OPT behave in exactly the same way (activate the same arm on each bandit invocation), we cannot guarantee that both will have the same total reward in the end. To enable meaningful comparison, we define a *payoff instance*, denoted by $I$, such that $I(i, n)$ denotes the reward generated by arm $i$ of bandit instance $S(n)$ for invocation $n$ in payoff instance $I$. The outcome of running BPOL or OPT on a given payoff instance is deterministic because the rewards are fixed in the payoff instance. Hence, we can compare BPOL and OPT on per payoff instance basis. Since each payoff instance arises with a certain probability, denoted as $\mathbb{P}(I)$, by taking expectation over all possible payoff instances of execution we can compare the expected performance of BPOL and OPT.

Let us consider invocation $n$ in payoff instance $I$. Let $B(I, n)$ and $O(I, n)$ denote the arms of bandit instance $S(n)$ activated under BPOL and OPT respectively. Based on the different possibilities that can arise, we classify invocation $n$ into one of three categories:

- *Category 1:* The arm activated by OPT, $O(I, n)$, is of smaller or equal expected reward in comparison to the arm activated by BPOL, $B(I, n)$. The expected reward of an arm is the product of its payoff probability and reward.

- *Category 2:* Arm $O(I, n)$ is of greater expected reward than $B(I, n)$, but $O(I, n)$ is not available for BPOL to activate at invocation $n$ due to budget restrictions.

- *Category 3:* Arm $O(I, n)$ is of greater expected reward than $B(I, n)$ and both arms $O(I, n)$ and $B(I, n)$ are available for BPOL to activate, but BPOL prefers to activate $B(I, n)$ over $O(I, n)$.

Let us denote the invocations of category $k$ (1, 2 or 3) by $\mathcal{N}^k(I)$ for payoff instance $I$. Let $bpol_k(N)$ and $opt_k(N)$ denote the expected reward obtained during the invocations of category $k$ (1, 2 or 3) by BPOL and OPT respectively. In [9] we show that

$$bpol_k(N) = \sum_{I \in \mathcal{I}} \left( \mathbb{P}(I) \cdot \sum_{n \in \mathcal{N}^k(I)} I(B(I, n), n) \right)$$

Similarly,

$$opt_k(N) = \sum_{I \in \mathcal{I}} \left( \mathbb{P}(I) \cdot \sum_{n \in \mathcal{N}^k(I)} I(O(I, n), n) \right)$$

Then for each $k$ we bound $opt_k(N)$ in terms of $bpol(N)$. In [9] we provide proof of each of the following bounds:

**Lemma 1** $opt_1(N) \leq bpol_1(N)$.

**Lemma 2** $opt_2(N) \leq bpol(N) + (|\mathcal{T}| \cdot r_{max})$, *where $|\mathcal{T}|$ denotes the number of arm types and $r_{max}$ denotes the maximum reward.*

**Lemma 3** $opt_3(N) = O(f(N))$.

From the above bounds we obtain our overall claim:

**Theorem 1** $bpol(N) \geq opt(N)/2 - O(f(N))$, *where $bpol(N)$ and $opt(N)$ denote the total expected reward obtained under BPOL and OPT respectively.*

**Proof:**
$$
\begin{aligned}
opt(N) \\
&= opt_1(N) + opt_2(N) + opt_3(N) \\
&\leq bpol_1(N) + bpol(N) + \big(|\mathcal{T}| \cdot r_{max}\big) + O(f(N)) \\
&\leq 2 \cdot bpol(N) + O(f(N))
\end{aligned}
$$
Hence, $bpol(N) \geq opt(N)/2 - O(f(N))$. ∎

If we supply MIX (Section 3) as input to our generic BPOL framework, we obtain *BMIX*, a policy for the budgeted unknown-CTR advertisement problem. Due to the way MIX structures and maintains its internal state, it is not necessary to restart a MIX instance when an advertiser's budget is depleted in BMIX, as specified in the generic BPOL framework (the exact steps of BMIX are given in [9]).

So far, for modeling purposes, we have assumed the search engine receives an entirely new batch of advertisements each day. In reality, ads may persist over multiple days. With BMIX, we can carry forward an ad's CTR estimate $(\hat{c}_{i,j})$ and display count $(n_{i,j})$ from day to day until an ad is revoked, to avoid having to re-learn CTR's from scratch each day. Of course the daily budgets reset daily, regardless of how long each ad persists. In fact, with a little care we can permit ads to be submitted and revoked at arbitrary times (not just at day boundaries). We describe this extension, as well as how we can incorporate and leverage prior beliefs about CTR's, in [9].

## 5 Experiments

From our general result of Section 4, we have a theoretical performance guarantee for BMIX. In this section we study BMIX empirically. In particular, we compare it with the greedy policy proposed for the known-CTR advertisement problem (Cells 1-IV in Figure 2). GREEDY displays the $C$ ads targeted for a query phrase that have the highest $(\hat{c}_{i,j} \cdot b_{i,j})$ values among the ads whose advertisers have enough remaining budgets; to induce a minimal amount of exploration, for an ad which has never been displayed before, GREEDY treats $\hat{c}_{i,j}$ as $\infty$ (our policies do this as well). GREEDY is geared exclusively toward *exploitation*. Hence, by comparing GREEDY with our policies, we can gauge the importance of *exploration*.

We also propose and evaluate the following variants of BMIX that we expect to perform well in practice:

**1. Varying the Exploration Factor.** Internally, BMIX runs instances of MIX to select which ads to display. As mentioned in Section 4, the priority function of MIX consists of an exploration factor $\big(\sqrt{\frac{2\ ln\ n_j}{n_{i,j}}}\big)$ and an exploitation factor $(c_{i,j})$. In [3] it was shown empirically that the following heuristical exploitation factor performs well, despite the absence of a known performance guarantee:

$$
\sqrt{\frac{ln\ n_j}{n_{i,j}} \cdot \min\left\{\frac{1}{4}, V_{i,j}(n_{i,j}, n_j)\right\}} \quad \text{where} \quad V_{i,j}(n_{i,j}, n_j) = \Big(\hat{c}_{i,j} \cdot (1 - \hat{c}_{i,j})\Big) + \sqrt{\frac{2\ ln\ n_j}{n_{i,j}}}
$$

Substituting this expression in place of $\sqrt{\frac{2\ ln\ n_j}{n_{i,j}}}$ in the priority function of BMIX gives us a new (heuristical) policy we call *BMIX-E*.

**2. Budget Throttling.** It is shown in [8] that in the presence of budget constraints, it is beneficial to display the ads of an advertiser less often as the advertiser's remaining budget decreases. In particular, they propose to multiply bids from advertiser $A_i$ by the following *discount factor*:

$$\phi(d_i') = 1 - e^{-d_i'/d_i}$$

where $d_i'$ is the current remaining budget of advertiser $A_i$ for the day and $d_i$ is its total daily budget. Following this idea we can replace $b_{i,j}$ by $\left(\phi(d_i') \cdot b_{i,j}\right)$ in the priority function of BMIX, yielding a variant we call *BMIX-T*. Policy *BMIX-ET* refers to use of heuristics 1 and 2 together.

## 5.1   Experiment Setup

We evaluate advertisement policies by conducting simulations over real-world data. Our data set consists of a sample of 85,000 query phrases selected at random from the Yahoo! query log for the date of February 12, 2006. Since we have the frequency counts of these query phrases but not the actual order, we ran the simulations multiple times with random orderings of the query instances and report the average revenue in all our experiment results. The total number of query instances is 2 million. For each query phrase we have the list of advertisers interested in it and the ads submitted by them to Yahoo!. We also have the budget constraints of the advertisers. Roughly 60% of the advertisers in our data set impose daily budget constraints.

In our simulation, when an ad is displayed, we decide whether a click occurs by flipping a coin weighted by the true CTR of the ad. Since true CTRs are not known to us (this is the problem we are trying to solve!), we took the following approach to assign CTRs to ads: from a larger set of Yahoo! ads we selected those ads that have been displayed more than thousand times, and therefore we have highly accurate CTR estimates. We regarded the distribution of these CTR estimates as the true CTR distribution. Then for each ad $a_{i,j}$ in the dataset we sampled a random value from this distribution and assigned it as CTR $c_{i,j}$ of the ad. (Although this method may introduce some skew compared with the (unknown) true distribution, it is the best we could do short of serving live ads just for the purpose of measuring CTRs).

We are now ready to present our results. Due to lack of space we consider a simple setting here where the set of ads is fixed and no prior information about CTR is available. We study the more general setting in [9].

## 5.2   Exploration/Exploitation Tradeoff

We ran each of the policies for a time horizon of ten days; each policy carries over its CTR estimates from one day to the next. Budget constraints are renewed each day. For now we fix the number of displayed ads ($C$) to 1. Figure 3 plots the revenue generated by each policy after a given number of days (for confidentiality reasons we have changed the unit of revenue). All policies (including GREEDY) estimate CTRs based on past observations, so as time passes by their estimates become more reliable and their performance improves. Note that the exploration factor of BMIX-E causes it to perform substantially better than that of BMIX. The budget throttling heuristic (BMIX-T and BMIX-ET) did not make much difference in our experiments.

All of our proposed policies perform significantly better than GREEDY, which underscores the importance of balancing exploration and exploitation. GREEDY is geared exclusively toward exploitation, so one might expect that early on it would outperform the other policies. However, that does not happen because GREEDY immediately fixates on ads that are not very profitable (*i.e.*, low $c_{i,j} \cdot b_{i,j}$).

Next we vary the number of ads displayed for each query ($C$). Figure 4 plots total revenue over ten days on the y-axis, and $C$ on the x-axis. Each policy earns more revenue when more ads are displayed (larger $C$). Our policies outperform GREEDY consistently across different values of C. In fact, GREEDY must display almost twice as many ads as BMIX-E to generate the same amount of revenue.

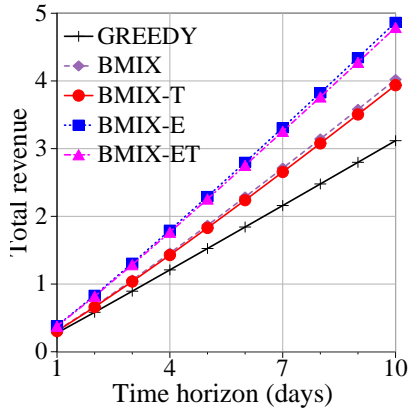

Figure 3: Revenue generated by different advertisement policies (C=1).

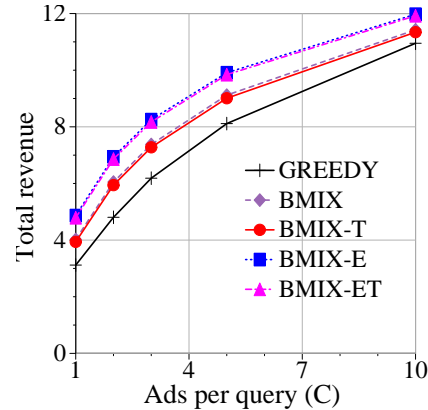

Figure 4: Effect of C (number of ads displayed per query).

## 6 Summary and Future Work

In this paper we studied how a search engine should select which ads to display in order to maximize revenue, when click-through rates are not initially known. We dealt with the underlying exploration/exploitation tradeoff using multi-armed bandit theory. In the process we contributed to bandit theory by proposing a new variant of the bandit problem that we call budgeted multi-armed multi-bandit problem (BMMP). We proposed a policy for solving BMMP and derived a performance guarantee. Practical extensions of our advertisement policies are given in the extended version of the paper. Extensive experiments over real ad data demonstrate substantial revenue gains compared to a greedy strategy that has no provision for exploration.

Several useful extensions of this problem can be conceived. One such extension would be to exploit similarity in ad attributes while inferring CTRs, as suggested in [7], instead of estimating the CTR of each ad independently. Also, an adversarial formulation of this problem merits study, perhaps leading to general consideration of how to manage exploration versus exploitation in game-theoretic scenarios.

## Footnotes

[1] The conventional multi-armed bandit problem is defined for $C = 1$. We generalize it to any $C \geq 1$ in this paper.

[2] We assume CTRs to be independent of one another.

## References

[1] N. Abe and A. Nakamura. Learning to Optimally Schedule Internet Banner Advertisements. In *ICML*, 1999.

[2] R. Agrawal. Sample Mean Based Index Policies with O(log n) Regret for the Multi-Armed Bandit Problem. *Advances in Applied Probability*, 27:1054–1078, 1995.

[3] P. Auer, N. Cesa-Bianchi, and P. Fischer. Finite-time Analysis of the Multi-Armed Bandit Problem. *Machine Learning*, 47:235–256, 2002.

[4] D. A. Berry and B. Fristedt. Bandit Problems: Sequential Allocation of Experiments. Chapman and Hall, London, 1985.

[5] N. Immorlica, K. Jain, M. Mahdian, and K. Talwar. Click Fraud Resistant Methods for Learning Click-Through Rates. In *WINE*, 2005.

[6] T. Lai and H. Robbins. Asymptotically Efficient Adaptive Allocation Rules. *Advances in Applied Mathematics*, 6:4–22, 1985.

[7] O. Madani and D. Decoste. Contextual Recommender Problems. In *Proceedings of the 1st International Workshop on Utility-based Data Mining*, 2005.

[8] A. Mehta, A. Saberi, U. Vazirani, and V. Vazirani. AdWords and Generalized On-line Matching. In *FOCS*, 2005.

[9] S. Pandey and C. Olston. Handling advertisements of unknown quality in search advertising, October, 2006. Technical report, available via `http://www.cs.cmu.edu/~spandey/publications/ctrEstimation.pdf`.

[10] P. Rusmevichientong and D. Williamson. An Adaptive Algorithm for Selecting Profitable Keywords for Search-Based Advertising Services. In *EC*, 2006.
